# Extreme Components Analysis

**Max Welling**
Department of Computer Science
University of Toronto
10 King's College Road
Toronto, M5S 3G5 Canada
*welling@cs.toronto.edu*

**Felix Agakov, Christopher K. I. Williams**
Institute for Adaptive and Neural Computation
School of Informatics
University of Edinburgh
5 Forrest Hill, Edinburgh EH1 2QL, UK
*{ckiw,felixa}@inf.ed.ac.uk*

## Abstract

Principal components analysis (PCA) is one of the most widely used techniques in machine learning and data mining. Minor components analysis (MCA) is less well known, but can also play an important role in the presence of constraints on the data distribution. In this paper we present a probabilistic model for "extreme components analysis" (XCA) which at the maximum likelihood solution extracts an optimal combination of principal and minor components. For a given number of components, the log-likelihood of the XCA model is guaranteed to be larger or equal than that of the probabilistic models for PCA and MCA. We describe an efficient algorithm to solve for the globally optimal solution. For log-convex spectra we prove that the solution consists of principal components only, while for log-concave spectra the solution consists of minor components. In general, the solution admits a combination of both. In experiments we explore the properties of XCA on some synthetic and real-world datasets.

## 1 Introduction

The simplest and most widely employed technique to reduce the dimensionality of a data distribution is to linearly project it onto the subspace of highest variation (principal components analysis or PCA). This guarantees that the reconstruction error of the data, measured with $L_2$-norm, is minimized. For some data distributions however, it is not the directions of large variation that are most distinctive, but the directions of very small variation, i.e. constrained directions. In this paper we argue that in reducing the dimensionality of the data, we may want to preserve these constrained directions alongside some of the directions of large variability.

The proposed method, termed "extreme components analysis" or XCA, holds the middle ground between PCA and MCA (minor components analysis–the method that projects on directions of low variability). The objective that determines the optimal combination of principal and minor components derives from the probabilistic formulation of XCA, which neatly generalizes the probabilistic models for PCA and MCA. For a fixed number of components, the XCA model will always assign higher probability to the (training) data than PCA or MCA, and as such be more efficient in encoding the data. We propose a very

simple and efficient algorithm to extract the optimal combination of principal and minor components and prove some results relating the shape of the log-spectrum to this solution.

The XCA model is inspired by Hinton's "product of experts" (PoE) model [1]. In a PoE, linear combinations of an input vector are penalized according to their negative log-probability and act as constraints. Thus, configurations of high probability have most of their constraints approximately satisfied. As we will see, the same is true for the XCA model which can therefore be considered as an under-complete product of Gaussians (PoG).

## 2 Variation vs. Constraint: PCA vs. MCA

Consider a plane embedded in 3 dimensions that cuts through the origin. There are 2 distinct ways to mathematically describe points in that plane:

$$\mathbf{x} = A\mathbf{y} \quad \forall \ \mathbf{y} \in \mathbb{R}^2, \qquad \text{or} \qquad \forall \ \mathbf{x} \in \mathbb{R}^3 \quad \text{s.t.} \ \mathbf{w}^T\mathbf{x} = 0 \tag{1}$$

where $A$ is a $3 \times 2$ matrix, the columns of which form a basis in the plane, and $\mathbf{w}$ is a vector orthogonal to the plane. In the first description we parameterize the modes of variation, while in the second we parameterize the direction of *no variation* or the direction in which the points are constrained. Note that we only need 3 real parameters to describe a plane in terms of its constraint versus 6 parameters to describe it in terms of its modes of variation. More generally, if we want to describe a $d$-dimensional subspace in $D$ dimensions we may use $D - d$ constraint directions or $d$ subspace directions.

Next consider the stochastic version of the above problem: find an accurate description of an approximately $d$-dimensional data-cloud in $D$ dimensions. The solution that probabilistic PCA (PPCA) [3, 4] provides is to model those $d$ directions using unit vectors $\mathbf{a}_i$ (organized as columns of a matrix $A$) while adding isotropic Gaussian noise in all directions,

$$\mathbf{x} = A\mathbf{y} + \mathbf{n} \qquad \mathbf{y} \sim \mathcal{N}[0, I_d] \qquad \mathbf{n} \sim \mathcal{N}[0, \sigma_0^2 I_D] \tag{2}$$

The probability density of $\mathbf{x}$ is Gaussian with covariance

$$C_{\mathbf{PCA}} = \langle \mathbf{x}\mathbf{x}^T \rangle = \sigma_0^2 I_D + AA^T. \tag{3}$$

In [4] it was shown that at the maximum likelihood solution the columns of $A$ are given by the first $d$ principal components of the data with length $||\mathbf{a}_i|| = \sqrt{\sigma_i^2 - \sigma_0^2}$ where $\sigma_i^2$ is the $i'th$ largest eigenvalue of the sample covariance matrix and $\sigma_0^2$ is equal to the average variance in the directions orthogonal to the hyperplane.

Alternatively, one may describe the data as $D - d$ approximately satisfied constraints, embedded in a high variance background model. The noisy version of the constraint $\mathbf{w}^T\mathbf{x} = 0$ is given by $z = \mathbf{w}^T\mathbf{x}$ where $z \sim \mathcal{N}[0, 1]$. The variance of the constrained direction, $1/||\mathbf{w}||^2$, should be smaller than that of the background model. By multiplying $D - d$ of these "Gaussian pancake" models [6] a probabilistic model for MCA results with *inverse* covariance given by,

$$C_{\mathbf{MCA}}^{-1} = \frac{I_D}{\sigma_0^2} + W^T W \tag{4}$$

where $\mathbf{w}^T$ form the rows of $W$. It was shown that at the maximum likelihood solution the rows of $W$ are given by the first $D - d$ *minor* components of the data with length $||\mathbf{w}_i|| = \sqrt{1/\sigma_i^2 - 1/\sigma_0^2}$ where $\sigma_i^2$ is the $i'th$ *smallest* eigenvalue of the sample covariance matrix and $\sigma_0^2$ is equal to the average variance in the directions orthogonal to the hyperplane. Thus, while PPCA explicitly models the directions of large variability, PMCA explicitly models the directions of small variability.

# 3 Extreme Components Analysis (XCA)

Probabilistic PCA can be interpreted as a low variance data cloud which has been stretched out in certain directions. Probabilistic MCA on the other hand can be thought of as a large variance data cloud which has been pushed inward in certain directions. Given the Gaussian assumption, the approximation that we make is due to the fact that we replace the variances in the remaining directions by their average. Intuitively, better approximations may be obtained by identifying the set of eigenvalues which, when averaged, induces the smallest error. The appropriate model, to be discussed below, will both have elongated and contracted directions in its equiprobable contours, resulting in a mix of principal and minor components.

## 3.1 A Probabilistic Model for XCA

The problem can be approached by either starting at the PPCA or PMCA model. The restricting aspect of the PPCA model is that the noise $\mathbf{n}$ is added in all directions in input space. Since adding random variables always results in increased variance, the directions modelled by the vectors $\mathbf{a}_i$ must necessarily have larger variance than the noise directions, resulting in principal components. In order to remove that constraint we need to add the noise only in the directions orthogonal to the $\mathbf{a}_i$'s. This leads to the following "causal generative model" model[1] for XCA,

$$\mathbf{x} = A\mathbf{y} + \mathcal{P}_A^\perp \mathbf{n} \qquad \mathbf{y} \sim \mathcal{N}[0, I_d] \qquad \mathbf{n} \sim \mathcal{N}[0, \sigma_0^2 I_D] \tag{5}$$

where $\mathcal{P}_A^\perp = I_D - A(A^T A)^{-1} A^T$ is the projection operator on the orthogonal complement of the space spanned by the columns of $A$. The covariance of this model is found to be

$$C_{\mathbf{XCA}} = \sigma_0^2 \mathcal{P}_A^\perp + AA^T. \tag{6}$$

Approaching the problem starting at the PMCA model we start with $d$ components $\{\mathbf{w}_i\}$ (organized as rows in $W$) and add isotropic noise to the remaining directions,

$$\mathbf{z}_1 = W\mathbf{x} \qquad \mathbf{z}_1 \sim \mathcal{N}[0, I_d] \qquad \mathbf{z}_2 = V\mathbf{x} \qquad \mathbf{z}_2 \sim \mathcal{N}[0, \sigma_0^2 I_{(D-d)}] \tag{7}$$

where the rows of $V$ form an orthonormal basis in the orthogonal complement of the space spanned by $\{\mathbf{w}_i\}$. Importantly, we will not impose any constraints on the norms of $\{\mathbf{w}_i\}$ or $\sigma_0$, i.e. the components are allowed to model directions of large or small variance. To derive the PDF we note that $(\{z_{1i}\}, \{z_{2i}\})$ are independent random variables implying that $P(\mathbf{z}_1, \mathbf{z}_2)$ is a product of marginal distributions. This is then converted to $P(\mathbf{x})$ by taking into account the Jacobian of the transformation $J_{(\mathbf{z}_1, \mathbf{z}_2) \to \mathbf{x}} = \sqrt{\det(WW^T)}$. The result is that $\mathbf{x}$ has a Gaussian distribution with with inverse covariance,

$$C_{\mathbf{XCA}}^{-1} = \frac{1}{\sigma_0^2} \mathcal{P}_W^\perp + W^T W \tag{8}$$

where $\mathcal{P}_W^\perp = I_D - W^T(WW^T)^{-1}W$ is the projection operator on the orthogonal complement of $W$. Also, $\det(C_{\mathbf{XCA}}^{-1}) = \det(WW^T)\sigma_0^{2(d-D)}$.

It is now not hard to verify that by identifying $A = W^\# \stackrel{\text{def}}{=} W^T(WW^T)^{-1}$ (the pseudo-inverse of $W$) the two models defined through eqns. 6 and 8 are indeed identical. Thus, by slightly changing the noise model, both PPCA and PMCA result in XCA (i.e. compare eqns.3,4,6,8).

## 3.2 Maximum Likelihood Solution

For a centered (zero mean) dataset $\{\mathbf{x}\}$ of size $N$ the log-likelihood is given by,

$$\mathcal{L} = -\frac{ND}{2}\log(2\pi) + \frac{N}{2}\log\det(WW^T) + \frac{N(D-d)}{2}\log\left(\frac{1}{\sigma_0^2}\right) - \frac{N}{2}\mathbf{tr}\left(C_{\mathbf{XCA}}^{-1}S\right) \quad (9)$$

where $S = \frac{1}{N}\sum_{i=1}^{N}\mathbf{x}_i\mathbf{x}_i^T \in \mathbb{R}^{D\times D}$ is the covariance of the data. To solve for the stationary points of $\mathcal{L}$ we take derivatives w.r.t $W^T$ and $1/\sigma_0^2$ and equate them to zero. Firstly, for $W$ we find the following equation,

$$W^\# - SW^T + \frac{1}{\sigma_0^2}\mathcal{P}_W^\perp SW^\# = 0. \quad (10)$$

Let $W^T = U\Lambda R^T$ be the singular value decomposition (SVD) of $W^T$, so that $U \in \mathbb{R}^{D\times d}$ forms an incomplete orthonormal basis, $\Lambda \in \mathbb{R}^{d\times d}$ is a full-rank diagonal matrix, and $R \in \mathbb{R}^{d\times d}$ is a rigid rotation factor. Inserting this into eqn. 10 we find,

$$U\Lambda^{-1}R^T - SU\Lambda R^T + \frac{1}{\sigma_0^2}(I_D - UU^T)SU\Lambda^{-1}R^T = 0. \quad (11)$$

Next we note that the projections of this equation on the space spanned by $W$ and its orthogonal complement should hold independently. Thus, multiplying equation 11 on the left by either $\mathcal{P}_W$ or $\mathcal{P}_W^\perp$, and multiplying it on the right by $R\Lambda^{-1}$, we obtain the following two equations,

$$U\Lambda^{-2} = UU^TSU, \quad (12)$$

$$SU\left(I_d - \frac{\Lambda^{-2}}{\sigma_0^2}\right) = UU^TSU\left(I_d - \frac{\Lambda^{-2}}{\sigma_0^2}\right). \quad (13)$$

Inserting eqn. 12 into eqn. 13 and right multiplying with $(I_d - \Lambda^{-2}/\sigma_0^2)^{-1}$ we find the eigenvalue equation[2],

$$SU = U\Lambda^{-2}. \quad (14)$$

Inserting this solution back into eqn. 12 we note that it is satisfied as well. We thus conclude that *U is given by the eigenvectors of the sample covariance matrix S, while the elements of the (diagonal) matrix $\Lambda$ are given by $\lambda_i = 1/\sigma_i$ with $\sigma_i^2$ the eigenvalues of S (i.e. the spectrum).*

Finally, taking derivatives w.r.t. $1/\sigma_0^2$ we find,

$$\sigma_0^2 = \frac{1}{D-d}\mathbf{tr}\left(P_W^\perp S\right) = \frac{1}{D-d}\left(\mathbf{tr}(S) - \mathbf{tr}(U\Lambda^{-2}U^T)\right) = \frac{1}{D-d}\sum_{i\in\mathcal{G}}\sigma_i^2 \quad (15)$$

where $\mathcal{G}$ is the set of all eigenvalues of $S$ which are *not* represented in $\Lambda^{-2}$. The above equation expresses the fact that these eigenvalues are being approximated through their average $\sigma_0^2$.

Inserting the solutions 14 and 15 back into the log-likelihood (eqn. 9) we find,

$$\mathcal{L} = -\frac{ND}{2}\log(2\pi e) - \frac{N}{2}\sum_{i\in\mathcal{C}}\log(\sigma_i^2) - \frac{N(D-d)}{2}\log\left(\frac{1}{D-d}\sum_{i\in\mathcal{G}}\sigma_i^2\right) \quad (16)$$

where $\mathcal{C}$ is the set of retained eigenvalues. The log-likelihood has now been reduced to a function of the discrete set of eigenvalues $\{\sigma_i^2\}$ of $S$.

### 3.3 An Algorithm for XCA

To optimize 16 efficiently we first note that the sum of the eigenvalues $\{\sigma_i^2\}$ is constant: $\sum_{i \in \mathcal{C} \cup \mathcal{G}} \sigma_i^2 = \mathbf{tr}(S)$. We may use this to rewrite $\mathcal{L}$ in terms of the retained eigenvalues only. We define the following auxiliary cost to be minimized which is proportional to $-\mathcal{L}$ up to irrelevant constants,

$$\mathcal{K} = \sum_{i \in \mathcal{C}} \log \sigma_i^2 + (D - d) \log(\mathbf{tr}(S) - \sum_{i \in \mathcal{C}} \sigma_i^2). \tag{17}$$

Next we recall an important result that was proved in [4]: *the minimizing solution has eigenvalues $\sigma_i^2, \ i \in \mathcal{G}$ which are contiguous in the (ordered) spectrum*, i.e. the eigenvalues which are averaged form a "gap" in the spectrum. With this result, the search for the optimal solution has been reduced from exponential to linear in the number of retained dimensions $d$. Thus we obtain the following algorithm for determining the optimal $d$ extreme components: (1) Compute the first $d$ principal components and the first $d$ minor components, (2) for all $d$ possible positions of the "gap" compute the cost $\mathcal{K}$ in eqn. 17, and (3) select the solution that minimizes $\mathcal{K}$.

It is interesting to note that the same equations for the log-likelihood ($\mathcal{L}$, eqn.16) and cost ($\mathcal{K}$, eqn.17) appear in the analysis of PPCA [4] and PMCA [6]. The only difference being that certain constraints forcing the solution to contain only principal or minor components are absent in eqn. 16. For XCA, this opens the possibility for mixed solutions with both principal and minor components. From the above observation we may conclude that *the optimal ML solution for XCA will always have larger log-likelihood on the training data then the optimal ML solutions for PPCA and PMCA*. Moreover, when XCA contains only principal (or minor) components, it must have equal likelihood on the training data as PPCA (or PMCA). In this sense XCA is the natural extension of PPCA and PMCA.

## 4 Properties of the Optimal ML Solution

We will now try to provide some insight into the nature of the the optimal ML solutions. First we note that the objective $\mathcal{K}$ is shifted by a constant if we multiply all variances by a factor $\sigma_i^2 \rightarrow \alpha \sigma_i^2$, which leaves its minima invariant. In other words, the objective is only sensitive to changing *ratios* between eigenvalues. This property suggests to use the logarithm of the eigenvalues of $S$ as the natural quantities since multiplying all eigenvalues with a constant results in a vertical shift of the log-spectrum. Consequently, the properties of the optimal solution only depend on the *shape* of the log-spectrum. In appendix A we prove the following characterization of the optimal solution,

**Theorem 1**
- *A log-linear spectrum has no preference for principal or minor components.*
- *The extreme components of log-convex spectra are principal components.*
- *The extreme components of log-concave spectra are minor components.*

Although a log-linear spectrum with arbitrary slope has no preference for principal or minor components, the slope does have an impact on the accuracy of the approximation because the variances in the gap are approximated by their average value. A spectrum that can be exactly modelled by PPCA with sufficient retained directions is one which has a pedestal, i.e. where the eigenvalues become constant beyond some value. Similarly PMCA can model exactly a spectrum which is constant and then drops off while XCA can model exactly a spectrum with a constant section at some arbitrary position. Some interesting examples of spectra can be obtained from the Fourier (spectral) representation of stationary Gaussian processes. Processes with power-law spectra $S(\omega) \propto \omega^{-\alpha}$ are log convex. An example of a spectrum which is log linear is obtained from the RBF covariance function

Table 1: Percent classification error of noisy sinusoids as a function of $g = D - d$.

| $g$ | 2 | 3 | 4 | 5 | 6 | 7 | 8 |
|---|---|---|---|---|---|---|---|
| $\epsilon_{XCA}$ | 1.88 | 1.91 | 2.35 | 1.88 | 2.37 | 3.27 | 28.24 |
| $\epsilon_{MCA}$ | 2.37 | 3.10 | 4.64 | 4.06 | 2.37 | 3.27 | 28.24 |
| $\epsilon_{PCA}$ | 1.88 | 2.50 | 12.21 | 14.57 | 19.37 | 32.99 | 30.14 |

with a Gaussian weight function, [7]. The RBF covariance function on the circle will give rise to eigenvalues $\lambda_i \propto e^{-\beta i^2}$, i.e. a log-concave spectrum.

Both PCA and MCA share the convenient property that a solution with $d$ components is contained in the solution with $d + 1$ components. This is not the case for XCA: the solution with $d + 1$ components may look totally different than the solution with $d$ components (see inset in Figure 1c), in fact they may not even share a single component!

## 5 Experiments

**Small Sample Effects**
When the number of data cases is small relative to the dimensionality of the problem, the log-spectrum tends to bend down on the MC side producing "spurious" minor components in the XCA solution. Minor components that result from finite sample effects, i.e. that do not exist in the infinite data limit, have an adverse effect on generalization performance. This is shown in Figure 1a for the "Frey-Faces" dataset, where we plot the log-likelihood for (centered) training and test data for both PCA and XCA. This dataset contains 1965 images of size $20 \times 28$, of which we used 1000 for training and 965 for testing. Since the number of cases is small compared to the number of dimensions, both PCA and XCA show a tendency to overfit. Note that at the point that minor components appear in the XCA solution ($d = 92$) the log-likelihood of the training data improves relative to PCA, while the log-likelihood of the test data suffers.

**Sinusoids in noise**
Consider a sum of $p$ sinusoids $Y(t) = \sum_{i=1}^{p} A_i \cos(\omega_i t + \phi_i)$ sampled at $D$ equally-spaced time points. If each $\phi_i$ is random in $(0, 2\pi)$ then the covariance $\langle Y(t) Y(t') \rangle = \sum_{i=1}^{p} P_i \cos \omega_i (t - t')$ where $P_i = A_i^2/2$. This signal defines a $2p$-dimensional linear manifold in the $D$-dimensional space (see [2] §12.5). By adding white noise to this signal we obtain a non-singular covariance matrix. Now imagine we have two such signals, each described by $p$ different powers and frequencies. Instead of using the exact covariance matrix for each we approximate the covariance matrix using either XCA, PMCA or PPCA. We then compare the accuracy of a classification task using either the exact covariance matrix, or the approximations. (Note that although the covariance can be calculated exactly the generating process is not in fact a Gaussian process.) By adjusting $p$, the powers and the frequencies of the two signals, a variety of results can be obtained. We set $D = 9$ and $p = 4$. The first signal had $P = (1.5, 2.5, 3, 2.5)$ and $\omega = (1.9, 3.5, 4.5, 5)$, and the second $P = (3, 2, 1.8, 1)$ and $\omega = (1.7, 2.9, 3.3, 5.3)$. The variance of the background noise was 0.5. Table 1 demonstrates error rates on 10000 test cases obtained for XCA, PMCA and PPCA using $g = D - d$ approximated components. For all values of $g$ the error rate for XCA is $\leq$ than that for PPCA and PMCA. For comparison, the optimal Gaussian classifier has an error rate of 1.87%. For $g = 2$ the XCA solution for both classes is PPCA, and for $g = 6, 7, 8$ it is PMCA; in between both classes have true XCA solutions. MCA behaviour is observed if $\sigma_0^2$ is low.

**2-D Positions of Face Features**
671 cases were extracted from a dataset containing 2-D coordinates of 6 features on frontal

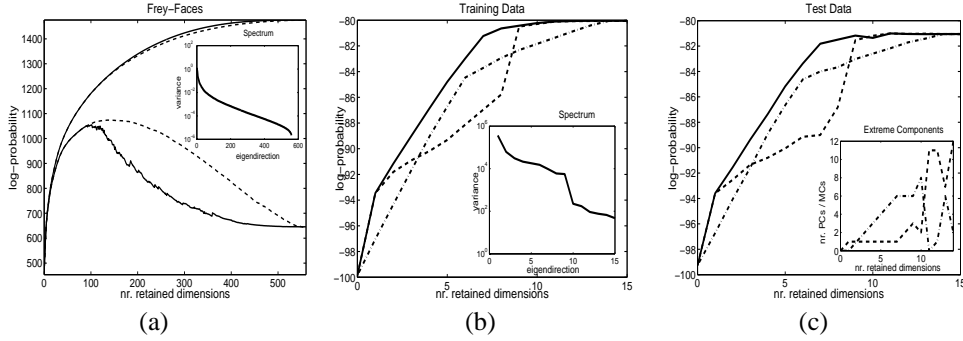

Figure 1: (a) Log-likelihood of the "Frey-faces" training data (top curves) and test data (bottom curves) for PCA (dashed lines) and XCA (solid lines) as a function of the number of components. Inset: log-spectrum of training data.(b) Log-likelihood of training data for PCA (dash), MCA (dash-dot) and XCA (solid) as a function of the number of components. Inset: log-spectrum of training data. (c) Log-likelihood of test data. Inset: number of PCs (dash) versus number of MCs (dash-dot) as a function of the number of components.

faces[3]. To obtain a translation and orientation invariant representation, we computed the 15 squared (Euclidean) distances between the features and removed their mean. In Figures 1b and 1c we show the log-likelihood for PCA, MCA and XCA of 335 training cases and 336 test cases respectively. Clearly, XCA is superior even on the test data. In the inset of Figure 1c we depict the number of PCs and MCs in the XCA solution as we vary the number of retained dimensions. Note the irregular behavior when the number of components is large.

## 6 Discussion

In this paper we have proposed XCA as the natural generalization of PCA and MCA for the purpose of dimensionality reduction. It is however also possible to consider a model with non-Gaussian components. In [5] the components were distributed according to a Student-t distribution resulting in a probabilistic model for undercomplete independent components analysis (UICA).

There are quite a few interesting questions that remain unanswered in this paper. For instance, although we have shown how to efficiently find the global maximum of the log-likelihood, we haven't identified the properties of the other stationary points. Unlike PPCA we expect many local maxima to be present. Also, can we formulate a Bayesian version of XCA where we predict the number and nature of the components supported by the data? Can we correct the systematic under-estimation of MCs in the presence of relatively few data cases? There are a number of extensions of the XCA model worth exploring: XCA with multiple noise models (i.e. multiple gaps in the spectrum), mixtures of XCA and so on.

## A Proof of Theorem 1

Using the fact that the sum and the product of the eigenvalues are constant we can rewrite the cost eqn.17 (up to irrelevant constants) in terms of the left-out eigenvalues of the spectrum only. We will also use the fact that the left-out eigenvalues are contiguous in the

spectrum, and form a "gap" of size $g \stackrel{\text{def}}{=} D - d$,

$$C = g \log \left( \sum_{i=i^*}^{i^*+g-1} e^{f_i} \right) - \sum_{i=i^*}^{i^*+g-1} f_i \tag{18}$$

where $f_i$ are the log-eigenvalues and $i^*$ is the location of the left hand side of the gap. We are interested in the change of this cost $\delta C$ if we shift it one place to the right (or the left). This can be expressed as

$$\delta C = g \log \left( 1 + \frac{e^{f_{i^*+g}} - e^{f_{i^*}}}{\sum_{i=i^*}^{i^*+g-1} e^{f_i}} \right) - (f(i^* + g) - f(i^*)). \tag{19}$$

Inserting a log-linear spectrum: $f_i = b + a \cdot i$ with $a < 0$ and using the result $\sum_{i=0}^{g-1} e^{a \cdot i} = (e^{ag} - 1)/(e^a - 1)$ we find that the change in $C$ vanishes for all log-linear spectra. This establishes the first claim. For the more general case we define corrections $c_i$ to the log-linear spectrum that runs through the points $f_{i^*}$ and $f_{i^*+g}$, i.e. $f_i = b + a \cdot i + c_i$. First consider the case of a convex spectrum between $i^*$ and $i^* + g$, which implies that all $c_i < 0$. Inserting this into 19 we find after some algebra

$$\delta C = g \log \left( 1 + \frac{e^{ag} - 1}{\sum_{i'=0}^{g-1} e^{a \cdot i' + c_{[i'+i^*]}}} \right) - ag. \tag{20}$$

Because all $c_i < 0$, the first term must be smaller (more negative) than the corresponding term in the linear case implying that $\delta C < 0$ (the second term is unchanged w.r.t the linear case). Thus, if the entire spectrum is log-convex the gap will be located on the right, resulting in PCs. A similar argument shows that for log-concave spectra the solutions consist of MCs only. In general log-spectra may have convex and concave pieces. The cost 18 is minimized when some of the $c_i$ are positive and some negative in such a way that, $\sum_{i'=0}^{g-1} e^{a \cdot i' + c_{[i'+i^*]}} \approx \sum_{i'=0}^{g-1} e^{a \cdot i'}$ Note that due to the exponent in this sum, positive $c_i$ have a stronger effect than negative $c_i$.

## Acknowledgements

We'd like to thank the following people for their invaluable input into this paper: Geoff Hinton, Sam Roweis, Yee Whye Teh, David MacKay and Carl Rasmussen. We are also very grateful to Pietro Perona and Anelia Angelova for providing the "feature position" dataset used in this paper.

## Footnotes

[1]Note however that the semantics of a two-layer directed graphical model is problematic since $p(\mathbf{x}|\mathbf{y})$ is improper.

[2]As we will see later, the left-out eigenvalues have to be contiguous in the spectrum, implying that the matrix $(I_d - \Lambda^{-2}/\sigma_0^2)^{-1}$ can only be singular if there is a retained eigenvalue that is equal to *all* left-out eigenvalues. This is clearly an uninteresting case, since the likelihood will not decrease if we leave this component out as well.

[3]The dataset was obtained by M. Weber at the computational vision lab at Caltech and contains the 2-D coordinates of 6 features (eyes, nose, 3 mouth features) of unregistered frontal face images.

## References

[1] G.E. Hinton. Products of experts. In *Proceedings of the International Conference on Artificial Neural Networks*, volume 1, pages 1–6, 1999.

[2] J.G. Proakis and D.G. Manolakis. *Digital Signal Processing: Principles, Algorithms and Applications*. Macmillan, 1992.

[3] S.T. Roweis. Em algorithms for pca and spca. In *Advances in Neural Information Processing Systems*, volume 10, pages 626–632, 1997.

[4] M.E. Tipping and C.M. Bishop. Probabilistic principal component analysis. *Journal of the Royal Statistical Society, Series B*, 21(3):611–622, 1999.

[5] M. Welling, R.S. Zemel, and G.E. Hinton. A tractable probabilistic model for projection pursuit. In *Proceedings of the Conference on Uncertainty in Artificial Intelligence*, 2003. accepted for publication.

[6] C.K.I. Williams and F.V. Agakov. Products of gaussians and probabilistic minor components analysis. *Neural Computation*, 14(5):1169–1182, 2002.

[7] H. Zhu, C. K. I. Williams, R. J. Rohwer, and M. Morciniec. Gaussian regression and optimal finite dimensional linear models. In C. M. Bishop, editor, *Neural Networks and Machine Learning*. Springer-Verlag, Berlin, 1998.
